# A Neural Network Approach for Three-Dimensional Object Recognition

**Volker Tresp**
Siemens AG, Central Research and Development
Otto-Hahn-Ring 6, D-8000 München 83
Germany

## Abstract

The model-based neural vision system presented here determines the position and identity of three-dimensional objects. Two stereo images of a scene are described in terms of shape primitives (line segments derived from edges in the scenes) and their relational structure. A recurrent neural matching network solves the correspondence problem by assigning corresponding line segments in right and left stereo images. A 3-D relational scene description is then generated and matched by a second neural network against models in a model base. The quality of the solutions and the convergence speed were both improved by using mean field approximations.

## 1 INTRODUCTION

Many machine vision systems and, to a large extent, also the human visual system, are model based. The scenes are described in terms of shape primitives and their relational structure, and the vision system tries to find a match between the scene descriptions and 'familiar' objects in a model base. In many situations, such as robotics applications, the problem is intrinsically 3-D. Different approaches are possible. Poggio and Edelman (1990) describe a neural network that treats the 3-D object recognition problem as a multivariate approximation problem. A certain number of 2-D views of the object are used to train a neural network to produce the standard view of that object. After training, new perspective views can be recognized.

In the approach presented here, the vision system tries to capture the true 3-D structure of the scene. Two stereo views of a scene are used to generate a 3-D

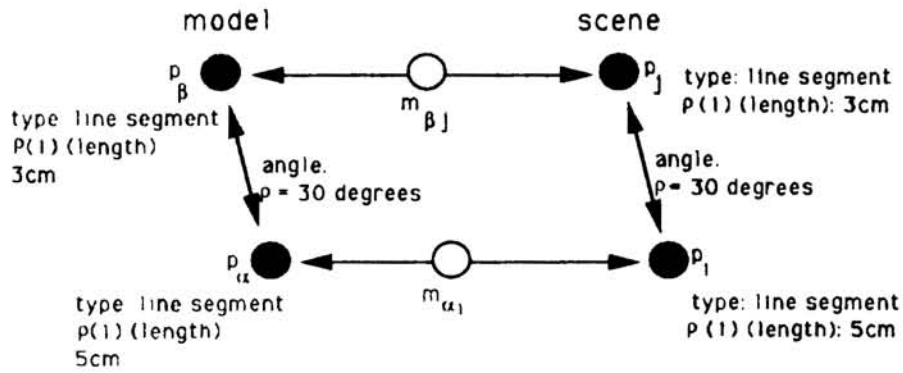

Figure 1: Match of primitive $p_\alpha$ to $p_i$.

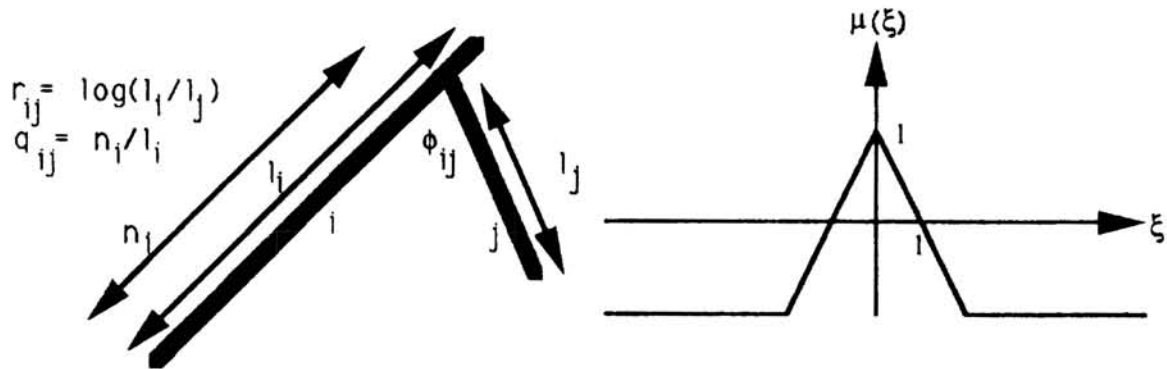

Figure 2: Definitions of $r$, $q$, and $\theta$ (left). The function $\mu()$ (right).

description of the scene which is then matched against models in a model base. The stereo correspondence problem and the model matching problem are solved by two recurrent neural networks with very similar architectures. A neuron is assigned to every possible match between primitives in the left and right images or, respectively, the scene and the model base. The networks are designed to find the best matches by obeying certain uniqueness constraints.

The networks are robust against the uncertainties in the descriptions of both the stereo images and the 3-D scene (shadow lines, missing lines). Since a partial match is sufficient for a successful model identification, opaque and partially occluded objects can be recognized.

## 2    THE NETWORK ARCHITECTURE

Here, a general model matching task is considered. The activity of a match neuron $m_{\alpha i}$ (Figure 1) represents the certainty of a match between a primitive $p_\alpha$ in the model base and $p_i$ in the scene description. The interactions between neurons can be derived from the network's energy function where the fixed points of the network correspond to the minima of the energy function. The first term in the energy

function evaluates the match between the primitives

$$E_P = -1/2 \sum_{\alpha i} \kappa_{\alpha i} m_{\alpha i}. \tag{1}$$

The function $\kappa_{\alpha i}$ is zero if the type of primitive $p_\alpha$ is not equal to the type of primitive $p_i$. If both types are identical, $\kappa_{\alpha i}$ evaluates the agreement between parameters $\rho_\alpha^p(k)$ and $\rho_i^p(k)$ which describe properties of the primitives. Here, $\kappa_{\alpha i} = \mu(\sum_k |\rho_\alpha^p(k) - \rho_i^p(k)|/\sigma_k^p)$ is maximum if the parameters of $p_\alpha$ and $p_i$ match (Figures 1 and 2).

The evaluation of the match between the relations of primitives in the scene and data base is performed by the energy term (Mjolsness, Gindi and Anadan, 1989)

$$E_S = -1/2 \sum_{\alpha,\beta,i,j} \chi_{\alpha,\beta,i,j} \, m_{\alpha i} m_{\beta j}. \tag{2}$$

The function $\chi_{\alpha i} = \mu(\sum_k |\rho_{\alpha,\beta}^r(k) - \rho_{i,j}^r(k)|/\sigma_k^r)$ is maximum if the relation between $p_\alpha$ and $p_\beta$ matches the relation between $p_i$ and $p_j$.

The constraint that a primitive in the scene should only match to one or no primitive in the model base (column constraint) is implemented by the additional (penalty-) energy term (Utans et al., 1989, Tresp and Gindi, 1990)

$$E_C = \sum_i [((\sum_\alpha m_{\alpha i}) - 1)^2 \sum_\alpha m_{\alpha i}]. \tag{3}$$

$E_C$ is equal to zero only if in all columns, the sum over the activations of all neurons is equal to one or zero and positive otherwise.

## 2.1 DYNAMIC EQUATIONS AND MEAN FIELD THEORY

### 2.1.1 $MFA_1$

The neural network should make binary decisions, match or no match, but binary recurrent networks get easily stuck in local minima. Bad local minima can be avoided by using an annealing strategy but annealing is time-consuming when simulated on a digital computer. Using a mean field approximation, one can obtain deterministic equations by retaining some of the advantages of the annealing process (Peterson and Söderberg, 1989). The network is interpreted as a system of interacting units in thermal contact with a heat reservoir of temperature $T$. Such a system minimizes the free energy $F = E - T\hat{S}$ where $\hat{S}$ is the entropy of the system. At $T = 0$ the energy $E$ is minimized. The mean value $v_{\alpha i} = < m_{\alpha i} >$ of a neuron becomes $v_{\alpha i} = 1/(1 + e^{-u_{\alpha i}/T})$ with $u_{\alpha i} = -\partial E/\partial v_{\alpha i}$. These equations can be updated synchronously, asynchronously or solved iteratively by moving only a small distance from the old value of $u_{\alpha i}$ in the direction of the new mean field.

At high temperatures $T$, the system is in the trivial solution $v_{\alpha i} = 1/2$ $\forall \alpha, i$ and the activations of all neurons are in the linear region of the sigmoid function. The system can be described by linearized equations. The magnitudes of all eigenvalues of the corresponding transfer matrix are less than 1. At a critical temperature $T_c$, the magnitude of at least one of the eigenvalues becomes greater than one and the trivial solution becomes unstable. $T_c$ and favorable weights for the different terms in the energy function can be found by an eigenvalue analysis of the linearized equations (Peterson and Söderberg, 1989).

### 2.1.2   $MFA_2$

The column constraint is satisfied by states with exactly one neuron or no neuron 'on' in every column. If only these states are considered in the derivation of the mean field equations, one can obtain another set of mean field equations, $v_{\alpha i} = 1 \times e^{u_{\alpha i}/T}/(1 + \sum_\beta e^{u_{\beta i}/T})$ with $u_{\alpha i} = -\partial E/\partial v_{\alpha i}$.

The column constraint term (Equation 3) drops out of the energy function and the energy surface in simplified. The high temperature fixed point corresponds to $v_{\alpha i} = 1/(N+1)$ $\forall \alpha, i$ where $N$ is the number of rows.

## 3   THE CORRESPONDENCE PROBLEM

To solve the correspondence problem, corresponding lines in left and right images have to be identified. A good assumption is that the appearance of an object in the left image is a distortion and shifted version of the appearance of the object in the right image with approximately the same scale and orientation. The machinery just developed can be applied if the left image is interpreted as the scene and the right image as the model.

Figure 3 shows two stereo images of a simple scene and the segmentation of left and right images into line segments which are the only primitives in this application. Lines correspond to the edges, structure and contours of the objects and shadow lines. The length of a line segment $\rho_i^p(1) = l_i$ is the descriptive parameter attached to each line segment $p_i$. Relations between line segments are only considered if they are in a local neighborhood: $\chi_{\alpha,\beta,i,j}$ is equal to zero if not both a) $p_\alpha$ is attached to line segment $p_\beta$ and b) line segment $p_i$ is attached to line segment $p_j$. Otherwise, $\chi_{\alpha,\beta,i,j} = \mu(|\phi_{\alpha\beta} - \phi_{ij}|/\sigma_\phi^r + |r_{\alpha\beta} - r_{ij}|/\sigma_r^r + |q_{\alpha\beta} - q_{ij}|/\sigma_q^r)$ where $\rho_{i,j}^r(1) = \phi_{ij}$ is the angle between line segments, $\rho_{i,j}^r(2) = r_{ij}$ the logarithm of the ratio of their lengths and $\rho_{i,j}^r(3) = q_{ij}$ the attachment point (Shumaker et al., 1989) (Figure 2).

Here, we have two uniqueness constraints: only at most one neuron should be active in each column or each row. The row constraint is enforced by an energy term equivalent to $E_C$: $E_R = \sum_\alpha [((\sum_i m_{\alpha i}) - 1)^2 \sum_i m_{\alpha i}]$.

## 4   DESCRIPTION OF THE 3-D OBJECT STRUCTURE

From the last section, we know which endpoints in the left image correspond to endpoints in the right image. If $D$ is the separation of both (in parallel mounted) cameras, $f$ the focal lengths of the cameras, $x_l, y_l, x_r, y_r$ the coordinates of a particular point in left and right images, the 3-D position of the point in camera coordinates $x, y, z$ becomes $z = Df/(x_r - x_l)$, $y = zy_r/f$, $x = zx_r/f + D/2$. This information is used to generate the 3-D description of the visible portion of the objects in the scene.

Knowing the true 3-D position of the endpoints of the line segments, the system concludes that the chair and the wardrobe are two distinct and spatially separated objects and that line segments 12 and 13 in the right image and 12 in the left image are not connected to either the chair or the wardrobe. On the other hand, it is not

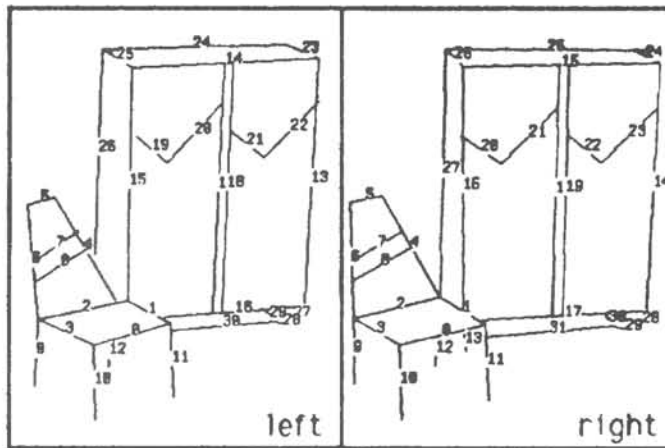
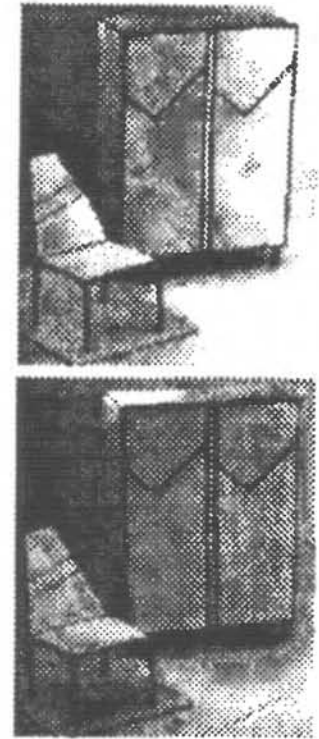

Figure 3: Stereo images of a scene and segmented images. The stereo matching network matched all line segments that are present in both images correctly.

obvious that the shadow lines under the wardrobe are not part of the wardrobe.

# 5   MATCHING OBJECTS AND MODELS

The scene description now must be matched with stored models describing the complete 3-D structures of the models in the data base. The model description might be constructed by either explicitly measuring the dimensions of the models or by incrementally assembling the 3-D structure from several stereo views of the models. Descriptive parameters are the (true 3-D) length of line segments $l$, the (true 3-D) angles $\phi$ between line segments and the (true 3-D) attachment points $q$. The knowledge about the 3-D structure allows a segmentation of the scene into different objects and the row constraint is only applied to neurons relating to the same object $O$ in the scene $E_{R'} = \sum_O \sum_\alpha [((\sum_{i \in O} m_{\alpha i}) - 1)^2 \sum_{i \in O} v_{\alpha i}]$.

Figure 4 shows the network after convergence. Except for the occluded leg, all line segments belonging to the chair could be matched correctly. All not occluded line segments of the wardrobe could be matched correctly except for its left front leg. The shadow lines in the image did not find a match.

# 6   3-D POSITION

In many applications, one is also interested in determining the positions of the recognized objects in camera coordinates. In general, the transformation between

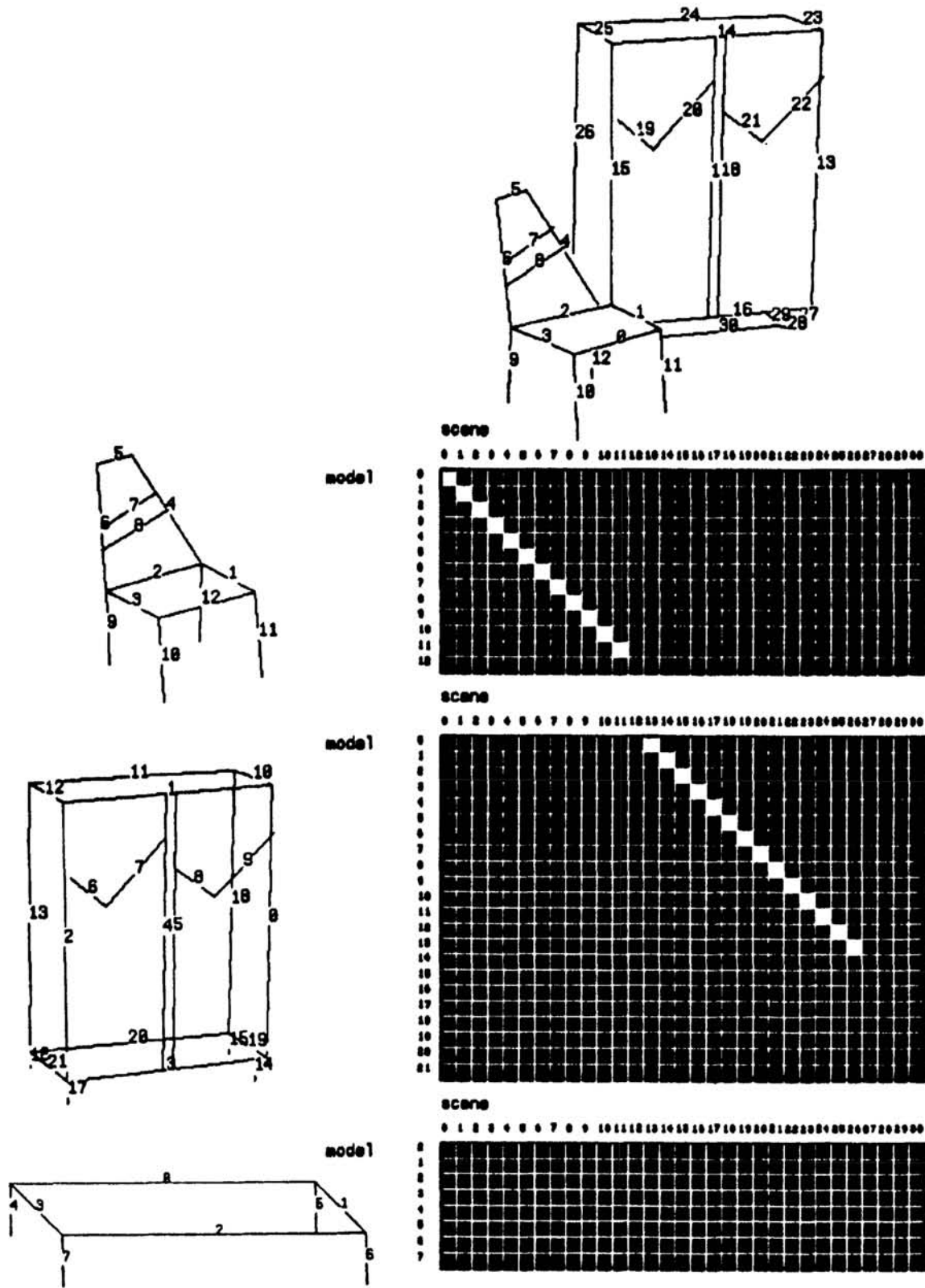

Figure 4: 3-D matching network.

an object in a standard frame of reference $X_0 = (x_0, y_0, z_0)$ and the transformed frame of reference $X_S = (x_s, y_s, z_s)$ can be described by $X_S = RX_0$, where $R$ is a $4 \times 4$ matrix describing a rotation followed by a translation. $R$ can be calculated if $X_0$ and $X_S$ are known for at least 4 points using, for example, the pseudo inverse or an ADALINE. Knowing the coefficients of $R$, the object position can be calculated. If an ADALINE is used, the error after convergence is a measure of the consistency of the transformation. A large error can be used as an indication that either a wrong model was matched, or certain primitives were misclassified.

# 7  DISCUSSION

Both $MFA_1$ and $MFA_2$ were used in the experiments. The same solutions were found in general, but due to the simpler energy surface, $MFA_2$ allowed greater time steps and therefore converged 5 to 10 times faster.

For more complex scenes, a hierarchical system could be considered. In the first step, simple objects such as squares, rectangles, and circles would be identified. These would then form the primitives in a second stage which would then recognize complete objects. It might also be possible to combine these two matching nets into one hierarchical net similar to the networks described by Mjolsness, Gindi and Anadan (1989).

## Acknowledgements

I would like to acknowledge the contributions of Gene Gindi, Eric Mjolsness and Joachim Utans of Yale University to the design of the matching network. I thank Christian Evers for helping me to acquire the images.

## References

Eric Mjolsness, Gene Gindi, P. Anadan. Neural Optimization in Model Matching and Perceptual Organization. *Neural Computation 1*, pp. 218-209, 1989.

Carsten Peterson, Bo Söderberg. A new method for mapping optimization problems onto neural networks. *International Journal of Neural Systems*, Vol. 1, No. 1, pp. 3-22, 1989.

T. Poggio, S. Edelman. A Network That Learns to Recognize Three-Dimensional Objects. *Nature*, No. 6255, pp. 263-266, January 1990.

Grant Shumaker, Gene Gindi, Eric Mjolsness, P. Anadan. Stickville: A Neural Net for Object Recognition via Graph Matching. Tech. Report No. 8908, Yale University, 1989.

Volker Tresp, Gene Gindi. Invariant Object Recognition by Inexact Subgraph Matching with Applications in Industrial Part Recognition. *International Neural Network Conference*, Paris, pp. 95-98, 1990.

Joachim Utans, Gene Gindi, Eric Mjolsness, P. Anadan. Neural Networks for Object Recognition within Compositional Hierarchies, Initial Experiments. Tech. Report No. 8903, Yale University, 1989.
